# An application of Reversible-Jump MCMC to multivariate spherical Gaussian mixtures

**Alan D. Marrs**
Signal & Information Processing Dept.
Defence Evaluation & Research Agency
Gt. Malvern, UK WR14 3PS
*marrs@signal.dra.hmg.gb*

## Abstract

Applications of Gaussian mixture models occur frequently in the fields of statistics and artificial neural networks. One of the key issues arising from any mixture model application is how to estimate the optimum number of mixture components. This paper extends the Reversible-Jump Markov Chain Monte Carlo (MCMC) algorithm to the case of multivariate spherical Gaussian mixtures using a hierarchical prior model. Using this method the number of mixture components is no longer fixed but becomes a parameter of the model which we shall estimate. The Reversible-Jump MCMC algorithm is capable of moving between parameter subspaces which correspond to models with different numbers of mixture components. As a result a sample from the full joint distribution of all unknown model parameters is generated. The technique is then demonstrated on a simulated example and a well known vowel dataset.

## 1 Introduction

Applications of Gaussian mixture models regularly appear in the neural networks literature. One of their most common roles in the field of neural networks, is in the placement of centres in a radial basis function network. In this case the basis functions are used to model the distribution of input data ($\mathbf{X}_i = [x_1, x_2, ..., x_d]^T$, ($i = 1, n$)), and the problem is one of mixture density estimation.

$$p(\mathbf{X}_i) = \sum_{j=1}^{k} \pi_j p(\mathbf{X}_i | \boldsymbol{\Theta}_\mathbf{j}), \qquad (1)$$

where $k$ is the number of mixture components, $\pi_j$ the weight or mixing proportion for component $j$ and $\Theta_j$ the component parameters (mean & variance in this case). The mixture components represent the basis functions of the neural network and their parameters (centres & widths) may be estimated using the expectation-maximisation (EM) algorithm.

One of the key issues arising in the use of mixture models is how to estimate the number of components. This is a model selection problem: the problem of choosing the 'correct' number of components for a mixture model. This may be thought of as one of comparing two (or more) mixture models with different components, and choosing the model that is 'best' based upon some criterion. For example, we might compare a two component model to one with a single component.

$$\mathbf{H_o} : p(\mathbf{X}_i | \boldsymbol{\Theta}) \;\; ; \;\; \mathbf{H_a} : \pi p(\mathbf{X}_i | \boldsymbol{\Theta}_1) + (1 - \pi)p(\mathbf{X}_i | \boldsymbol{\Theta}_2). \qquad (2)$$

This may appear to be a case of testing of nested hypotheses. However, it has been noted [5] that the standard frequentist hypothesis testing theory (generalised likelihood ratio test) does not apply to this problem because the desired regularity conditions do not hold. In addition, if the models being tested have 2 and 3 components respectively, they are not strictly nested. For example, we could equate any pair of components in the three component model to the components in the two component model, yet how do we choose which component to 'leave out'?

## 2  Bayesian approach to Gaussian mixture models

A full Bayesian analysis treats the number of mixture components as one of the parameters of the model for which we wish to find the conditional distribution. In this case we would represent the joint distribution as a hierarchical model where we may introduce prior distributions for the model parameters, ie.

$$p(k, \pi, z, \boldsymbol{\Theta}, \mathbf{X}) = p(k)p(\pi|k)p(z|\pi, k)p(\boldsymbol{\Theta}|z, \pi, k)p(\mathbf{X}|\boldsymbol{\Theta}, z, \pi, k), \qquad (3)$$

where $\pi = (\pi_j)_{j=1}^{k}$, $\boldsymbol{\Theta} = (\boldsymbol{\Theta}_j)_{j=1}^{k}$ and $z = (z_i)_{i=1}^{n}$ are allocation variables introduced by treating mixture estimation as a hidden data problem with $z_i$ allocating the $i$th observation to a particular component. A simplified version of this model can be derived by imposing further conditional independencies, leading to the following expression for the joint distribution

$$p(k, \pi, z, \boldsymbol{\Theta}, \mathbf{X}) = p(k)p(\pi|k)p(z|\pi, k)p(\boldsymbol{\Theta}|k)p(\mathbf{X}|\boldsymbol{\Theta}, z). \qquad (4)$$

In addition, we add an extra layer to the hierarchy representing priors on the model parameters giving the final form for the joint distribution

$$p(\lambda, \delta, \eta, k, \pi, z, \boldsymbol{\Theta}, \mathbf{X}) = p(\lambda)p(\delta)p(\eta)p(k|\lambda)p(\pi|k, \delta)p(z|\pi, k) \times$$
$$p(\boldsymbol{\Theta}|k, \eta)p(\mathbf{X}|\boldsymbol{\Theta}, z). \qquad (5)$$

Until recently a full Bayesian analysis has been mathematically intractable. Model comparison was carried out by conducting an extensive search over all possible

model orders comparing Bayes factors for all possible pairs of models. What we really desire is a method which will estimate the model order along with the other model parameters. Two such methods based upon Markov Chain Monte Carlo (MCMC) techniques are reversible-jump MCMC [2] and jump-diffusion [3].

In the following sections, we extend the reversible-jump MCMC technique to multivariate spherical Gaussian mixture models. Results are then shown for a simulated example and an example using the Peterson-Barney vowel data.

## 3   Reversible-jump MCMC algorithm

Following [4] we define the priors for our hierarchical model and derive a set of 5 move types for the reversible jump MCMC sampling scheme. To simplify some of the MCMC steps we choose a prior model where the prior on the weights is Dirichlet and the prior model for $\mu_j = [\mu_{j_1}, ..., \mu_{j_d}]^T$ and $\sigma_j^{-2}$ is that they are drawn independently with normal and gamma priors,

$$\pi \sim D(\delta, ..., \delta) \ , \ \mu_j \sim N(\eta, A^{-1}) \ , \ \sigma_j \sim \Gamma(\alpha, \beta), \tag{6}$$

where for the purposes of this study we follow[4] and define the hyper-parameters thus: $\delta = 1.0$; $\eta$ is set to be the mean of the data; $A$ is the diagonal precision matrix for the prior on $\mu_j$ with components $a_j$ which are taken to be $1/r_j^2$ where $r_j$ is the data range in dimension $j$; $\alpha = 2.0$ and $\beta$ is some small multiple of $1/r_j^2$.

The moves then consist of: **I**: updating the weights; **II**: updating the parameters $(\mu, \sigma)$; **III**: updating the allocation; **IV**: updating the hyper-parameters; **V**: splitting one component into two, or combining two into one.

The first 4 moves are relatively simple to define, since the conjugate nature of the priors leads to relatively simple forms for the full conditional distribution of the desired parameter. Thus the first 4 moves are Gibbs sampling moves and the full conditional distributions for the weights $\pi_j$, means $\mu_j$, variances $\sigma_j$ and allocation variables $z_i$ are given by:

$$p(\pi_j|...) \sim D(\delta + n_1, ..., \delta + n_k), \tag{7}$$

where $n_k$ is the number of observations allocated to component $k$;

$$p(\mu_j|...) = \prod_{m=1}^{d} p(\mu_{j_m}|...) : p(\mu_{j_m}|...) \sim N(\frac{n_j \overline{x}_{i_m} \sigma_j^{-2} + a_m \eta_m}{(n_j \sigma_j^{-2} + a_m)}, (n_j \sigma_j^{-2} + a_m)^{-1}), \tag{8}$$

where we recognise that $\mu_j$ is an $d$ dimensional vector with components $\mu_{j_m} (m = 1, d)$, $\eta_m$ are the components of the $\mu_j$ prior mean and $a_m$ represent the diagonal components of $A$.

$$p(\sigma_j^{-2}|...) = \Gamma(\nu + n_j - 1, \frac{1}{2} \sum_{i=1:z_{ij}=1}^{n} (X_i - \mu_j)^T (X_i - \mu_j) + \beta); \tag{9}$$

and

$$p(z_i = j|...) \propto \frac{\pi_j}{\sigma_j} \exp\left(- \sum_{m=1}^{d} \frac{(x_i - \mu_{j_m})^2}{\sigma_j^2}\right). \tag{10}$$

The final move involves splitting/combining model components. The main criteria which need to be met when designing these moves are that they are irreducible, aperiodic, form a reversible pair and satisfy detailed balance[1]. The MCMC step for this move takes the form of a Metropolis-Hastings step where a move from state $y$ to state $y'$ is proposed, with $\pi(y)$ the target probability distribution and $q_m(y,y')$ the proposal distribution for the move $m$. The resulting move is then accepted with probability $\alpha_m$

$$\alpha_m = min\left\{1, \frac{\pi(y')q_m(y',y)}{\pi(y)q_m(y,y')}\right\}. \tag{11}$$

In the case of a move from state $y$ to a state $y'$ which lies in a higher dimensional space, the move may be implemented by drawing a vector of continuous random variables $u$, independent of $y$. The new state $y'$ is then set using an invertible deterministic function of $x$ and $u$. It can be shown [2] that the acceptance probability is then given by

$$\alpha_m = min\left\{1, \frac{\pi(y')r_m(y')}{\pi(y)r_m(y)q(u)}|\frac{\partial y'}{\partial(y,u)}|\right\}, \tag{12}$$

where $r_m(y)$ is the probability of choosing move type $m$ when in state $y$, and $q(u)$ is the density function of $u$.

The initial application of the reversible jump MCMC technique to normal mixtures [4] was limited to the univariate case. This yielded relatively simple expressions for the split/combine moves, and, most importantly, the determinant of the Jacobian of the transformation from a model with $k$ components to one with $k+1$ components was simple to derive. In the more general case of multivariate normal models care must be taken in prescribing move transformations. A complicated transformation will lead to problems when the |Jacobian| for a $d$-dimensional model is required.

For multivariate spherical Gaussian models, we randomly choose a model component from the current $k$ component model. The decision is then made to split or combine with one of its neighbours with probability $p_{s_k}$ and $p_{c_k}$ respectively (where $p_{c_k} = 1 - p_{s_k}$). If the choice is to combine the component, we label the chosen component $z_1$, and choose $z_2$ to be a neighbouring component $j$ with probability $\propto 1/r_j$ where $r_j$ is the distance from the component $z_1$. The new component resulting from the combination of $z_1$ and $z_2$ is labelled $z_c$ and its parameters are calculated from:

$$\pi_{z_c} = \pi_{z_1} + \pi_{z_2} \;\; ; \;\; \mu_{z_{c_{j:(j=1,d)}}} = \frac{(\pi_{z_1}\mu_{z_{c1_j}} + \pi_{z_2}\mu_{z_{c_j}})}{(\pi_{z_1}+\pi_{z_1})};$$
$$\sigma_{z_c}^2 = \frac{(\pi_{z_1}\sigma_{z_1}^2 + \pi_{z_2}\sigma_{z_2})}{(\pi_{z_1}+\pi_{z_2})}. \tag{13}$$

If the decision is to split, the chosen component is labelled $z_c$ and it is used to define two new model components $z_1$ and $z_2$ with weights and parameters conforming to (13). In making this transformation there are $2+d$ degrees of freedom, so we need to generate $2+d$ random numbers to enable the specification of the new component parameters. The random numbers are denoted $u_1$, $\mathbf{u}_2 = [u_{2_1},...,u_{2_d}]^T$ and $u_3$. All are drawn from $Beta(2,2)$ distributions while the components of $\mathbf{u}_2$ each have probability 0.5 of being negative. The split transformation is then defined by:

$$\pi_{z_1} = u_1\pi_{z_c} \;\; , \;\; \pi_{z_2} = (1-u_1)\pi_{z_c}$$
$$\mu_{z_{1_j}} = \mu_{z_{c_j}} - u_{2_j}\sigma_{z_c}\sqrt{\frac{\pi_{z_2}}{\pi_{z_1}}} \;\; , \;\; \mu_{z_{2_j}} = \mu_{z_{c_j}} + u_{2_j}\sigma_{z_c}\sqrt{\frac{\pi_{z_1}}{\pi_{z_2}}}$$

$$\sigma_{z_1}^2 = u_3 \sigma_{z_c}^2 \frac{\pi_{z_2}}{\pi_{z_1}} \quad , \quad \sigma_{z_2}^2 = (1 - u_3)\sigma_{z_c}^2 \frac{\pi_{z_1}}{\pi_{z_2}}. \tag{14}$$

Once the new components have been defined it is necessary to evaluate the probability of choosing to combine component $z_1$ with component $z_2$ in this new model.

Having proposed the split/combine move all that remains is to calculate the Metropolis-Hastings acceptance probability $\alpha$, where $\alpha = min(1, R)$ for the split move and $\alpha = min(1, 1/R)$ for the combine move. Where in the case of a split move from a model with $k$ components to one with $k + 1$ components, or a combine move from $k + 1$ to $k$, R is given by:

$$
\begin{aligned}
R = \quad & \frac{\prod_{i=1:z_{ij}=z_1}^{n} p(\mathbf{X}_i | \boldsymbol{\Theta}, c) \prod_{i=1:z_{ij}=z_2}^{n} p(\mathbf{X}_i | \boldsymbol{\Theta}, c)}{\prod_{i=1:z_{ij}=z_c}^{n} p(\mathbf{X}_i | \boldsymbol{\Theta}, c)} \times \\
& \frac{\pi_{z_1}^{\delta-1+n_1} \pi_{z_2}^{\delta-1+n_2}}{\pi_{z_c}^{\delta-1+n_1+n_2} B(\delta, k\delta)} \times \\
\prod_{m=1}^{d} \sqrt{\frac{a_m}{(2\pi)}} & \exp\left(-\frac{1}{2} a_m \left((\mu_{z_{1m}} - \eta_m)^2 + (\mu_{z_{2m}} - \eta_m)^2 - (\mu_{z_{cm}} - \eta_m)\right)\right) \times \\
& \frac{\beta^\alpha}{\Gamma(\alpha)} \left(\frac{\sigma_{z_c}^2}{\sigma_{z_1}^2 \sigma_{z_2}^2}\right)^{(\alpha-1)} \exp\left(-\beta(\sigma_{z_1}^{-2} + \sigma_{z_2}^{-2} - \sigma_{z_c}^{-2})\right) \times \\
& \frac{p_{c_{k+1}}}{p_{s_k} p_{alloc}} \left(g_{2,2}(u_1) g_{1,1}(u_3) \prod_{j=1}^{d} g_{2,2}(u_2,)\right) \times \\
& \frac{\pi_{z_c} \sigma_{z_c}^{d+1}}{(2((1-u_1)u_1)^{(d+1)/2} \sqrt{(1-u_3)u_3}}, \tag{15}
\end{aligned}
$$

where $g_{2,2}()$ denotes a *Beta(2, 2)* density function. The first line on the R.H.S is due to the ratio of likelihoods for those observations assigned to the components in question, the subsequent three lines are due to the prior ratios, the fifth line is due to the the proposal ratio and the last line due to the |Jacobian| of the transformation. The term $p_{alloc}$ represents a combination of the probability of obtaining the current allocation of data to the components in question and the probability of choosing to combine components $z_1$ and $z_2$.

## 4  Results

To assess this approach to the estimation of multivariate spherical Gaussian mixture models, we firstly consider a toy problem where 1000 bivariate samples were generated from a known 20 component mixture model. This is followed by an analysis of the Peterson-Barney vowel data set comprising 780 samples of the measured amplitude of four formant frequencies for 10 utterances. For this mixture estimation example, we ignore the class labels and consider the straight forward density estimation problem.

### 4.1  Simulated data

The resulting reversible-jump MCMC chain of model order can be seen in figure 1, along with the resulting histogram (after rejecting the first 2000 MCMC samples). The histogram shows that the *maximum a posteriori* value for model order is 17. The MAP estimate of model parameters was obtained by averaging all the 17 component model samples, the estimated model is shown in figure 2 alongside the original generating model. The results are rather encouraging given the large number of model components and the relatively small number of samples.

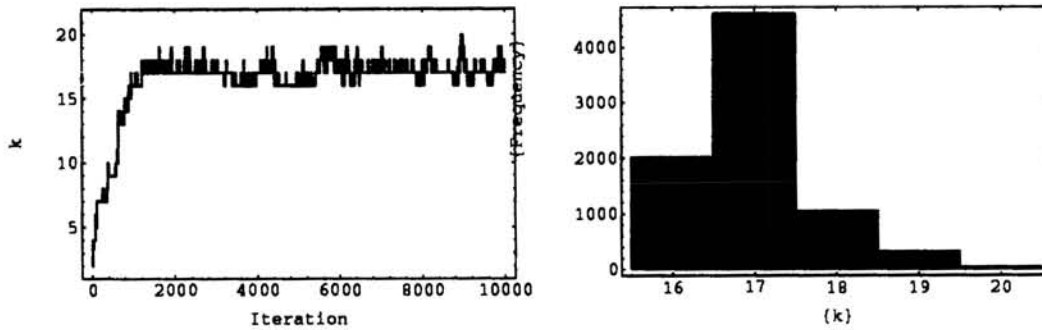

Figure 1: Reversible-jump MCMC chain and histogram of model order for simulated data.

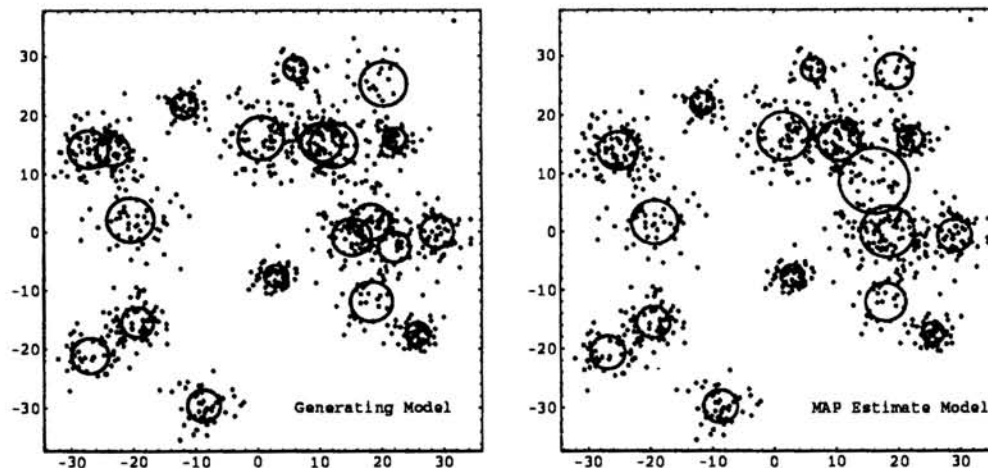

Figure 2: Example of model estimation for simulated data.

## 4.2 Vowel data

The reversible-jump MCMC chain of model order for the Peterson-Barney vowel data example is shown in figure 3, alongside the resulting MAP model estimate. For ease of visualisation, the estimated model and data samples have been projected onto the first two principal components of the data. Again, the results are encouraging.

## 5 Conclusion

One of the key problems when using Gaussian mixture models is estimation of the optimum number of components to include in the model. In this paper we extend the reversible-jump MCMC technique for estimating the parameters of Gaussian mixtures with an unknown number of components to the multivariate spherical Gaussian case. The technique is then demonstrated on a simulated data example and an example using a well known dataset.

The attraction of this approach is that the number of mixture components is not fixed at the outset but becomes a parameter of the model. The reversible-jump MCMC approach is then capable of moving between parameter subspaces which

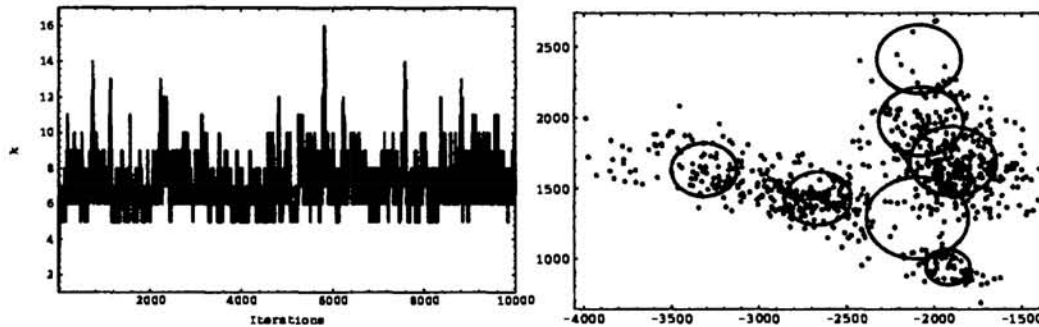

Figure 3: Reversible-jump MCMC chain of model order and MAP estimate of model (projected onto first two principal components) for vowel data.

correspond to models with different numbers of mixture components. As a result a sample of the full joint distribution is generated from which the posterior distribution for the number of model components can be derived. This information may then either be used to construct a Bayesian classifier or to define the centres in a radial basis function network.

# References

[1] W.R. Gilks, S. Richardson, and D.J. Spiegelhalter Eds. *Markov Chain Monte Carlo in Practice*. Chapman and Hall, 1995.

[2] P.J. Green. Reversible jump MCMC computation and Bayesian model determination. *Boimetrika*, 82:711–732, 1995.

[3] D.B. Phillips and A.F.M. Smith. Bayesian model comparison via jump diffusions. In W.R. Gilks, S. Richardson, and D.J. Spiegelhalter, editors, *Markov Chain Monte Carlo in Practice*. Chapman and Hall, 1995.

[4] S. Richardson and P. J. Green. On Bayesian analysis of mixtures with an unknown number of components. *J. Royal Stat. Soc. Series B*, 59(4), 1997.

[5] D.M. Titterington, A.F.M. Smith, and U.E. Makov. *Statistical Analysis of Finite Mixture Distributions*. Wiley, 1985.

